# Why are some word orders more common than others? A uniform information density account

**Luke Maurits, Amy Perfors & Daniel Navarro**
School of Psychology,
University of Adelaide,
Adelaide, South Australia, 5000
{luke.maurits, amy.perfors, daniel.navarro}@adelaide.edu.au

## Abstract

Languages vary widely in many ways, including their canonical word order. A basic aspect of the observed variation is the fact that some word orders are much more common than others. Although this regularity has been recognized for some time, it has not been well-explained. In this paper we offer an information-theoretic explanation for the observed word-order distribution across languages, based on the concept of Uniform Information Density (UID). We suggest that object-first languages are particularly disfavored because they are highly non-optimal if the goal is to distribute information content approximately evenly throughout a sentence, and that the rest of the observed word-order distribution is at least partially explainable in terms of UID. We support our theoretical analysis with data from child-directed speech and experimental work.

## 1 Introduction

Many of the world's languages are sensitive to word order. In these languages, the order in which words are spoken conveys a great deal of the sentence's meaning. The classic English example is the distinction between "dog bites man" and "man bites dog", which differ in terms of who is biting whom. The so-called "basic" word order of a language is defined according to the order of three of the principal components of basic transitive sentences: subject (S), verb (V) and object (O). This results in six logically distinct word orders: SOV, SVO, VSO, VOS, OVS and OSV (e.g., English has SVO basic word order). Curiously, the world's order-sensitive languages make use of these six possibilities in an uneven fashion. According to a survey of 402 languages [17], the majority of languages are either SOV (44.78%) or SVO (41.79%). VSO (9.20%) is much less frequent but still significant, and very few languages make use of VOS (2.99%), OVS (1.24%) or OSV (0.00%) as their basic word order. Broadly speaking, the basic pattern appears to be (SOV, SVO) > VSO > (VOS, OVS) > OSV. This non-uniformity is a striking empirical finding that demands some explanation. Unfortunately, most of the explanations that have been offered are either proximate explanations that simply shift the question, or else are circular.

One of the most straightforward explanations is that the observed word order frequencies may be the consequence of genetically encoded biases toward particular orders, as part of the universal grammar hypothesis; this possibility is considered in [4]. However, this can be only a proximate explanation: *why* does our genetic endowment happen to bias us in the particular way that it does? And if there is nothing special about the observed distribution – if it is not an adaption to the environment – why have thousands of years of adaption and genetic drift not blurred it into something closer to uniformity?

A similar objection can be made against the proposal that all languages which are alive today descend from a single common ancestor, and that this proto–language used SOV word order [8], ex-

plaining the observation that SOV is the most common word order today. If there is nothing special about SOV, why has random drift (this time in language evolution, not human genetic evolution) not more significantly changed the word order distribution from its ancient form? Furthermore, it is clear that ancient SOV languages must have changed into SVO languages much more frequently into than, say, VOS languages in order to arrive at the current state of affairs. Common descent from SOV cannot explain this by itself.

Another explanation seeks to derive word order frequencies as a consequence of more fundamental or general linguistic principles. Three such principles are presented in [17]: the "theme-first principle", "verb-object bonding" and the "animate-first principle". These principles do an excellent job of explaining the observed word order frequencies; the frequency of each word order is proportional to the number of the principles which that word order permits to be realized (all three principles are realized in SOV and SVO, two are realized in VSO, one in VOS and OVS, and none in OSV). However, these principles are primarily motivated by the fact that a large body of cross-linguistic data is consistent with them. Without a deeper justification, they are, in essence, a useful recharacterization of the data; to offer them as explanations of patterns in that data is circular. In other words, it is not clear *why* these principles work.

In this paper we propose a novel explanation for the observed distribution of word orders across languages, based on uniform information density (UID). The UID hypothesis [13, 10] suggests that language producers unconsciously endeavor to keep the rate of information transmission as close to constant as possible when speaking. We use the term "information" here in its information-theoretic sense of reduction of entropy (uncertainty) of a random variable (where the random variable is the underlying meaning of an utterance). Conveying information via speech with a uniform information density represents an optimal solution to the computational problem of conveying information over a noisy channel in a short time with low probability of error. A listener's comprehension of an utterance is made more difficult if a syllable, word or clause which carries a lot of information is lost due to ambient noise or problems with articulation or perception. The most error resistant strategy is therefore to convey minimal information with each unit of speech. Unfortunately, this leads to other problems – namely, that it will take excessive time to convey any meaningful quantity of information. The best trade off between time efficiency and error resistance is to spread information content as equally as possible across units and have each unit carry as much information as it can without exceeding the threshold for error correctability (the channel capacity). Also, UID minimizes the difficulty involved in online sentence processing, assuming that the difficulty of processing a speech unit increases superlinearly with that unit's surprisal [13].

The UID hypothesis is supported by a range of empirical evidence. It suggests that speakers should attempt to slow down the rate at which information is conveyed when unexpected, high entropy content is being discussed, and increase the rate when predictable, low entropy content is being discussed. This prediction is supported by findings indicating that certain classes of words [1] and syllables [3] are spoken more slowly in unexpected contexts. In addition, analysis of corpus data suggests that the entropy of sentences taken out of context is higher for sentences further into a body of text [7, 12]. Furthermore, the use of both optional contractions (e.g., "you are" vs. "you're") [2] and optional function words in relative clauses (e.g., "how big is the house **that** you live in?" vs. "how big is the house you live in?") [14, 11] appears to be affected by information density considerations, with contractions used less often when the relative clause is unexpected.

We propose that the basic word order of a language influences the average uniformity of information density for sentences in that language, and that a preference for languages that are closer to the UID ideal can explain some of the structure in the observed distribution over basic word orders. The layout of the rest of the paper is as follows. In Section 2 we describe the underlying conceptual model and terminology using a simple illustrative example. In Section 3,

## 2  Development of hypothesis and illustrative examples

This work is based on a simple probabilistic model of language production. We assume that languages are grounded in a *world*, consisting of *objects* (elements of a set $\mathcal{O}$) and *actions* (which are binary relations between objects, and elements of a set $\mathcal{R}$, such that if $r \in \mathcal{R}$ then $r \subset \mathcal{O} \times \mathcal{O}$). An *event* in the world is a triple consisting of a relation $r$ and two objects $o_1, o_2$ and is written $(o_1, r, o_2)$. Events in the world are generated probabilistically in a sequential fashion, as independent identically

distributed draws from a probability distribution $P$ over the set of events $\mathcal{O} \times \mathcal{R} \times \mathcal{O}$. We assume that a language consists of nouns (each of which corresponds to a unique object) and verbs (each of which corresponds to a unique action). Utterances are generated from events by combining the three relevant words in one of the six possible orders. Each utterance is therefore three words long (there are no function words in the model). This defines a probabilistic generative model for three-word utterances.

To make this idea more concrete, we construct a simple toy world consisting of thirteen objects and two relations. Five of the objects represent individual people (ALICE, BOB, EVE, MALLORY, TRENT) and the other eight represent items which are either food (APPLE, BREAD, CAKE, RICE) or drink (COFFEE, COLA, JUICE, WATER). The two relations are EAT and DRINK, so that the events in this world represent particular people eating or drinking particular items (e.g. (ALICE, DRINK, COFFEE)). Impossible events (e.g., (COFFEE, DRINK, ALICE)) are given zero probability in the event distribution $P$. A diagrammatic representation of all the non-zero probabilities of $P$ is available in the supplementary material, but the salient features of the example are as follows: each of the five people eat and drink equally often, and equally as often as each other; nobody drinks foods or eats drinks; and each person has their own particular idiosyncratic distribution over which foods they prefer to eat and which drinks they prefer to drink.

What is the link between word order and information density in this toy world? Consider a listener who learns about events in this toy world by hearing three-word utterances (such as "Alice eats apples" or "Bob drinks coffee"), one word at a time. Until they have heard all three words in the utterance, there will generally remain some degree of uncertainty about what the event is, with the uncertainty decreasing as each word is heard. Formally, the event underlying an utterance is a random variable, and the listener's uncertainty is represented by the entropy of that random variable.

Before any words are spoken, the observer's uncertainty is given by the entropy of the event distribution (which we refer to as the *base entropy* and denote $H_0$):

$$H_0 = H(P) = \sum_{(o_1, r, o_2)} -P(o_1, r, o_2) \log(P(o_1, r, o_2)), \tag{1}$$

where the sum is taken over all possible events in the world. After the first word, the observer's uncertainty about the event is reduced, and now corresponds to the entropy of one of the conditional distributions, $P(o_1, o_2|r), P(r, o_2|o_1)$ or $P(o_1, r|o_2)$, depending on whether the first word corresponds to the action (VSO or VOS word order), the person (SVO or SOV word order) or the food/drink (OVS or OSV word order). Similarly, after the second word, the uncertainty is the entropy of one of the conditional distributions $P(o_2|o_1, r), P(o_1|r, o_2)$ or $P(r|o_1, o_2)$, depending again on word order. After the third word the event is uniquely determined and the entropy is zero.

This means that for any particular event, the six different choices of word order each define a different monotonically decreasing sequence of intermediate entropies, with the first point in the sequence always being $H_0$ and the final point always being zero. Equivalently, the different choices of word order result in different distributions of the total information content of a sentence amongst its constituent words. We call sequences of entropies $(H_0, H_1, H_2, 0)$ *entropy trajectories*, and sequences of information $(I_1 = H_0 - H_1, I_2 = H_2 - H_1, I_3 = H_2)$ *information profiles*. Figure 1 shows the entropy trajectories and corresponding information profiles for the event (ALICE, EAT APPLE) in our toy world, for three different word orders. The figure demonstrates the correspondence between trajectories and profiles, as well as the dependency of both on word order. Note that in the figure we have normalized entropies and informations, so that $H_0 = 1$.

If we make the simplifying assumption that all words are of equal length[1], the UID hypothesis suggests that the ideal shape of an entropy trajectory is a perfectly straight line from the initial base entropy to the eventual zero entropy, or, equivalently, that the ideal shape of an information profile is for each word to convey one third of the total information. Figure 1 demonstrates that some trajectories are better realizations of this ideal than others. For example, in our toy world the entropy trajectories for the word orders SOV, OSV and OVS (two of which are pictured in Figure 1) are perfectly horizontal at various points (equivalently, some words carry zero information) because

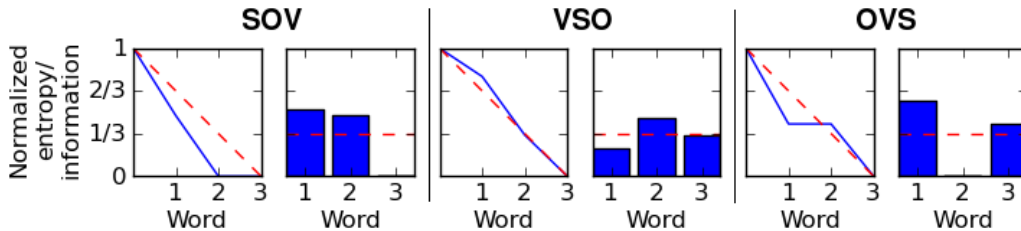

Figure 1: The entropy trajectories and corresponding information profiles for the event (ALICE, EAT, APPLE) in our toy world, for three different word orders. Dotted lines indicate the ideal trajectory and profile according to the UID hypothesis. Observe that word orders in which the object preceeds the verb have significant "troughs" in their information profiles, making them far from ideal. This pattern arises because of the event structure in our toy world; our question is what word orders are optimal given real-world event structure.

knowledge of the object in this world uniquely determines the verb (since foods are strictly eaten and drinks are strictly drunk). Thus, any word order that places O before V renders the verb entirely uninformative, in significant conflict with the UID hypothesis.

To formalize the intuitive notion of distance from the UID ideal we define the *UID deviation score* $D(I)$ of any given information profile $I = (I_1, I_2, I_3)$. $D(I)$ is given by the formula:

$$D(I) = \frac{3}{4} \sum_{i=1}^{3} \left| \frac{I_i}{H_0} - \frac{1}{3} \right|. \tag{2}$$

It is easy to verify that the UID ideal information profile, with $I_1 = I_2 = I_3$, has a deviation score of zero, and the least-ideal profile, in which all information is conveyed by a single word, has a deviation score of 1.

The UID deviation score allow us, for each event in the model world, to produce both an ordering of the word orders from "most UID-like" to "least UID-like", as well as a quantitative measure of the extent to which each word order approaches uniform information density. We can straightforwardly calculate a mean deviation score for the entire model world, by summing the scores for each individual event and weighting by that event's probability according to the event distribution $P$. This lets us assess the extent to which each word order is UID-suited to a given world. For our toy world, the ordering of word orders from lowest to highest mean deviation score is: VSO, VOS, SVO, OVS, SOV, OSV.

Of course, our toy world is a highly contrived example, and so there is no reason to expect it to produce the observed cross-linguistic distribution of word orders. This is because we constructed the artificial $P$ distribution to be pedagogically useful, not to reflect the real-world distribution of events. The toy example is intended only as a demonstration of the core idea underlying our hypothesis: that different choices of word order map the same probabilistic structure of the world ($P$) onto different information profiles. Since these profiles have differing levels of information density uniformity, the UID hypothesis implies a preference ranking of word orders.

What are the mean deviation scores when the event distribution $P$ more accurately approximates reality? Does the preferred ranking of word orders implied by the UID hypothesis reflect the observed cross-linguistic distribution of word orders? We investigate these questions in the rest of the paper.

## 3 Corpus analysis

Our work above implies that a particular word ordering in a language is good to the extent that it produces minimal UID deviation scores for events in the world. Accordingly, it would be ideal to assess the optimality of a particular word ordering with respect to the true distribution over "psychologically meaningful" events in the everyday environment. Although we do not have access to this distribution, we may be able to construct sensible approximations. One option is to assume that spontaneous speech is informative about event probabilities – that the probability with which speakers discuss an event is roughly proportional to the actual frequency or psychological importance of that event. Guided by this assumption, in this section we estimate $P$ on the basis of child-directed

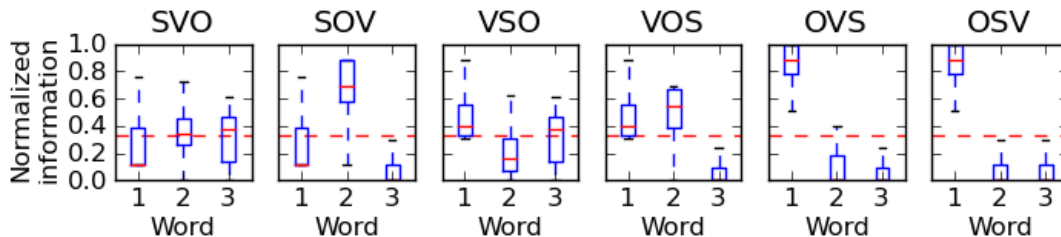

Figure 2: Distribution of information across words for the world instantiated from an English corpus

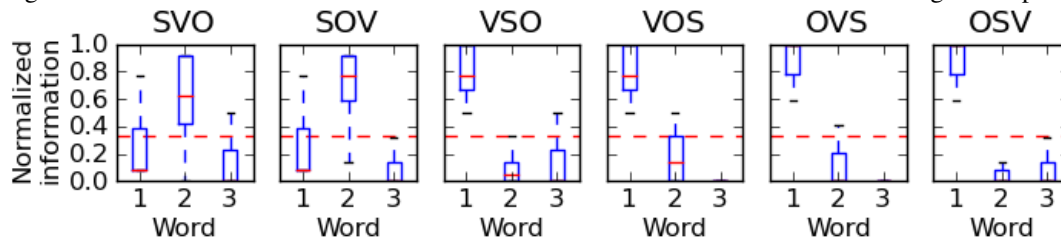

Figure 3: Distribution of information across words for the world instantiated from a Japanese corpus

speech corpora in two languages, English and Japanese. We use child-directed speech even though the UID hypothesis applies equally well to adult speakers for two reasons: because child-directed speech is more amenable to the particular analysis we provide (which requires relatively simple sentences), and because children learn their language's basic word very quickly and accurately [9, 5], suggesting that any aspect of primary linguistic data relevant to word order learning must be present in simple child-directed speech.

As our source of English data, we take the "Adam" transcripts from the Brown corpus [5] in the CHILDES database [15]. From this data we extract all of the child-directed utterances involving a random subset of the singly transitive verbs in the corpus (a total of 544 utterances). The subjects and objects of these utterances define the set $\mathcal{O}$ and the verbs define the set $\mathcal{R}$. In our analysis, we treat each utterance as a distinct event, setting the probability of an event in $P$ to be proportional to the number of times the corresponding utterance occurs in the corpus. Thus the event distribution $P$ is a measure of the probability that speakers of the language choose to discuss events (rather than their frequency in the real world). For simplicity, we ignore adjectives, plurality, tense, and so forth: for instance, the utterances "the black cat sat on the mat" and "the cats are sitting on the soft mat" would both be mapped to the same event, (CAT, SIT, MAT). Utterances involving pronouns which were considered likely to refer to a wide range of objects across the corpus (such as "it", "this", etc.) were discarded, while those involving pronouns which in the context of the discourse could be expected to refer to a small set of objects (such as "he" or "she") were retained.

Figure 2 shows the distribution of information amongst words (summarizing all of the model world's information profiles) for all six word orders according to the event distribution $P$ derived from the "Adam" transcripts. The mean deviation scores for the six word orders are (from lowest to highest) VSO (0.38), SVO (0.41), VOS (0.48), SOV (0.64), OSV (0.78), OVS (0.79).

To guard against the possibility that these results are a by-product of the fact that English has basic word order SVO, we repeat the method discussed above using utterances involving singly transitive verbs taken from the "Asato", "Nanami" and "Tomito" transcripts in the MiiPro corpus of the CHILDES database, which is in Japanese (basic order SOV). From these transcripts we retreive 134 utterances. The distribution of information amongst words for the event distribution derived from the Japanese transcripts are shown in Figure 3. The mean deviation scores are SVO (0.66), VSO (0.71), SOV (0.72), VOS (0.72), OSV (0.82), OVS (0.83). This is not precisely the ranking recovered from the English corpus, but there are clear similarities, which we discuss later.

## 4 Experiment

In the previous analyses, the event distribution $P$ was estimated on the basis of linguistic input. While this is sensible in many respects, it blurs the distinction between the frequency of events in

Table 1: Objects and relations in our experiment's model world. Asterisks denote "actor" status.

| Objects | APPLE, BEAR*, BED, BELLY-BUTTON, BLANKET, BUNNY*, CAT*, CHAIR, CHEESE, COOKIE, COW*, CRACKER, CUP, DIAPER, DOOR, DUCK*, EAR, FISH*, FLOWER, FOOT*, HAIR, HAND*, HAT, HORSE*, KEY*, LIGHT, MILK, MOUTH*, NOSE*, OUTSIDE, PERSON*, PIG*, SPOON*, TV, TELEPHONE, TOE*, TOOTH*, TREE, WATER |
|---|---|
| Relations | BITE, DRINK, EAT, HELP, HUG, KISS, OPEN, READ, SEE, SWING |

Table 2: Most and least probable completions of event frames according to experimentally determined event distribution $P$

| Event frame | Most probable completion | Least probable completion |
|---|---|---|
| PERSON EAT _________ | APPLE | DOOR |
| CAT DRINK _________ | MILK | BED |
| PERSON _________ CAT | HELP | EAT |
| _________ EAT FLOWER | COW | TOOTH |

the world and the frequency with which speakers choose to discuss those events. In one version of the UID hypothesis, we would expect that word order would be optimal with respect to the latter, "speaker-weighted" frequencies. We refer to this as the "weak" hypothesis since it only requires that a language be "internally" consistent, insofar as the word order is expected to be optimal with respect to the topics spoken about. However, there is also a "strong" version of the hypothesis, which states that the language must also be optimal with respect to the perceived frequencies of events in the external world. To test the strong version of the UID word order hypothesis, it is not valid to rely on corpus analysis. Accordingly, in this section we present the results of an experiment designed to measure people's perceptions regarding which events are most likely.

Our experiment consists of three parts. In the first part we identify the objects $\mathcal{O}$ and relations $\mathcal{R}$ for the model world based on the first words learned by English-speaking children, on the assumption that those words would reflect the objects and relations that are highly salient. The MacArthur Communicative Development Inventory [6] provides a list of those words, along with norms for when they are learned. We identified all of the words that were either singly-transitive verbs or nouns that were potential subjects or objects for these verbs, yielding 324 nouns and 81 verbs. The only transformation we made to this list was to replace all nouns that referred to specific people (e.g., "Mommy" or "Grandpa") with a single noun "Person". In order to limit the total number of possible events to a number tractable for parts two and three of the experiment, we then identified the 40 objects and 10 relations[2] uttered by the highest percentage of children below the age of 16 months; these comprise the sets $\mathcal{O}$ and $\mathcal{R}$. The objects and relations are shown in Table 1.

The 40 objects and 10 relations in our world define a total of 16,000 events, but the overwhelming majority of the events in the world are physically impossible (e.g., (TELEVISION, DRINK, CAT)) and thus should receive a probability of 0. The goal of the second part of the experiment was to identify these impossible events. The first step was to identify the subset of objects capable of acting as actors, indicated with asterisks in Table 1. We set the probability of events whose subjects were non-actors to zero, leaving 6,800 events. To identify which of these events were still impossible, we had two participants[3] judge the possibility or impossibility of each, obtaining two judgements for each event. When both judges agreed that an event was impossible, its probability was set to zero; if they disagreed, we solicited a third judgement and set the event probability to zero if the majority agreed that it was impossible. At the end of this process, a total of 2,536 events remained. Subsequent analysis revealed that many participants had interpreted the noun OUTSIDE as an adverb in events such as (BEAR, EAT, OUTSIDE), leading to events which should properly

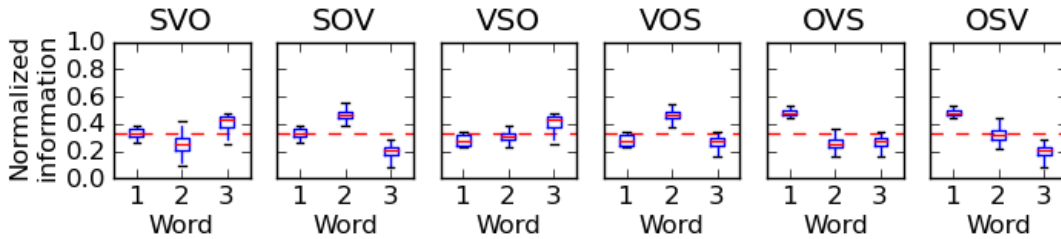

Figure 4: Distribution of information across words for the world instantiated from the experimentally produced event distribution.

have been considered impossible being classed as possible; we therefore set all events involving the noun OUTSIDE which did not involve the verb SEE to also be impossible. This reduced the number of events to 2,352.

In the final part of the experiment, we derived a probability distribution over the remaining, possible events using the responses of participants to a large number of judgement tasks. In each task, participants were presented with a pair of events and asked to indicate which of the two events they considered most probable. Full details of this part of the experiment are available in the supplementary material. Table 2 shows the most and least probable completions of several event frames according to the distribution $P$ produced by our experiment. The completions are in line with common sense, although some of the least probable completions are in fact physically impossible (e.g. (CAT, DRINK, BED)), suggesting that the filtering in part two was not quite perfect.

We now analyse the $P$ distribution we have estimated. The distribution of information among words is shown in Figure 4 and the mean deviation scores are VSO (0.17), SVO (0.18), VOS (0.20), SOV (0.23), OVS (0.23), OVS (0.24).

## 5 Discussion

On the basis of two corpora of child-directed speech, in different languages, and an experiment, we have derived three different event distributions which are assumed to represent the important features of the probabilistic structure of the physical world. From these different distributions we derive three different preferential rankings of word orders according to the UID hypothesis. From the English corpus, we get VSO > SVO > VOS > SOV > OSV > OVS; from the Japanese corpus, we get SVO > VSO > SOV = VOS > OSV > OVS; from the experiment, we get VSO > SVO > VOS > SOV = OVS > OSV. While these three rankings are not in perfect agreement, there is some degree of common structure. All three rankings are compatible with the partial ranking (SVO, VSO) > (SOV, VOS) > (OVS, OSV). How does this compare with the empirically observed ranking (SOV, SVO) > VSO > (VOS, OVS) > OVS?

The strongest empirical regularity regarding word order frequency - that object-first word orders are extremely rare - coincides with our most robust finding: object-first word orders lead to the least uniform information density in all three of our estimated event distributions. These orders together account for less than 2% of the world's word order-sensitive languages, and in all our models have deviation scores that are notably greater than the deviation scores of the other word orders. What is the reason for this effect? As the profiles in Figures 2, 3 and 4 indicate, object-first word orders deviate from uniformity because the first word (the object) carries disproportionate amount of information. This seems to occur because many objects are predictive of very few subjects or verbs. For instance, hearing the object word "water" implies only a few possibilities for verbs (e.g., "drink"), which in turn restricts the subjects (e.g. to living things). By contrast, hearing the verb "drink" implies many possibilities for objects (e.g., "water", "coffee", "cola", "juice", etc.).

There are further points of agreement between the rankings produced by our analyses and the empirical data. All three of our estimated event distributions lead to word order rankings in which VSO is ranked more highly than VOS, which is in agreement with the data. In fact, in all of our rankings, SVO and VSO occupy the two highest positions (though their relative position varies), consistent with the fact that these word orders occupy the second and third highest positions in the empirical

ranking respectively, and are two of the only three word orders which appear with any appreciable frequency.

The greatest apparent discrepancy between the rankings produced by our analyses and the empirical data is the fact that SOV word order, which occurs frequently in real languages, appears to be only moderately compatible with the UID hypothesis. One possible explanation for this is that some other factor besides UID-compatibility has influenced the distribution of word orders, and this factor may favour SOV sufficiently to lift it to the top or equal-top place in a combined ranking. Another possibility is to combine the idea we saw earlier of common descent from SOV with the idea that word order change away from SOV is influenced by the UID hypothesis. This explanation could also lift SOV word order to a higher position in the word order ranking.

To what extent are our rankings consistent with the the theme-first principle (TFP), verb-object bonding (VOB) and animate-first principle (AFP) principles of [17], which perfectly explain the empirical ranking? The three orders that permit the greatest realization of the TFP and AFP principles are SOV, SVO, and VSO. We note that two of these orders, SVO and VSO, are consistently ranked highest in our results, and the third, SOV, is typically not too far behind. In fact, with the event distribution derived from the Japanese corpus, SOV is in equal third place with VOS. This suggests that perhaps the UID word order hypothesis is unable to provide a complete explanation of all of the word order rankings, but *is* able provide a sensible justification for the TFP and/or AFP.

A full consideration of the effects of word order on information density should not limit itself only to the considerations made in this paper, and so our results here must be considered only preliminary. For instance, we have given no consideration to sentences involving intransitive verbs (SV sentences), sentences without an explicit subject (VO sentences), or sentences involving ditransitive verbs ($SVO_1O_2$ sentences). A word order optimal for one of these sentence classes may not be optimal for others, so that the question of how to meaningfully combine the results of separate analyses becomes a central challenge in such an extended study. Furthermore, a number of other word order parameters beyond basic word order may have a significant effect on information density, such as whether a language uses prepositions or postpositions, or the relative position of nouns and adjectives or nouns and relative clauses. For instance, consider the order of nouns and adjectives. The utterance "I ate the..." can be completed by any edible object, but "I ate the red..." only by those objects which are both edible *and* red. Thus, adjectives which preceed unexpected nouns can be used to "smooth out" what might otherwise be sudden spikes in information density. Adjectives which come after nouns cannot do this. Several correlations and rules are known to exist between various word order parameters, and it is possible that these effects may be able to be explained in terms of information density.

On the whole, while the word order rankings recovered from our analyses do not perfectly match the empirically observed ranking, they are in much better agreement with observation than one would expect if a preference for UID had played no role whatsoever. Furthermore, the particular pattern of what our rankings do and do not explain, and the ways our two rankings differ, are consistent with a weaker hypothesis that UID may be able to provide a principled cognitive explanation for the theme-first and/or animate-first principles of earlier work. It is possible that the discrepancies which do exist between our results and the empirical distribution could be explained by a combination of more and richer data and consideration of additional word order parameters. It is also the case that even if information theoretic concerns *have* exerted a significant influence on language evolution, there is no reason to expect them to have been the *only* such influence: genetic and social factors as well additional cognitive constraints may have played some role as well, so that the UID hypothesis alone need not explain *all* the observed regularity. Regardless, we have shown that information-theoretic principles can explain several aspects of the empirical distribution of word orders, and most robustly explains the most pronounced of these aspects: the nearly complete lack of object-first languages. Moreover, they do so on independently justified, general cognitive principles, and as such represent a significant advance in our understanding of word order.

# 6   Acknowledgements

DJN was supported by an Australian Research Fellowship (ARC grant DP-0773794). Kirsty Maurits assisted significantly in the translation of utterances from the Japanese transcripts.

## Footnotes

[1]Obviously this is not true. However, in order for this simplifying assumption to skew our results, the length of nouns would need to vary systematically depending on the relative frequency with which the nouns were the subject and orbject of sentences, which is highly unlikely to be the case.

[2]The ratio of 4 objects for every 1 relation was chosen to reflect the proportion of each reported in [6].

[3]This experiment involved 11,839 binary decisions in the second part and 35,280 binary choices in the third part. In order to collect such a large quantity of data in a reasonable time period, we used Amazon.com's "Mechanical Turk" web application to distribute the judgement tasks to a large international pool of participants, who completed the tasks using their web browsers in exchange for small payments of cash or Amazon.com store credit. A total of 8,956 participants contributed in total, presumably but not verifiably representing a broad range of nationalities, ages, levels of education, etc.

## References

[1] Alan Bell, Daniel Jurafsky, Eric Fosler lussier, Cynthia Girand, Michelle Gregory, and Daniel Gildea. Effects of disfluencies, predictability, and utterance position on word form variation in English conversation. *Journal of the Acoustical Society of America*, 113(2), 2003.

[2] Austin F. Frank and T. Florian Jaeger. Speaking Rationally: Uniform Information Density as an Optimal Strategy for Language Production. In *Proceedings of the 30th Annual Meeting of the Cognitive Science Society*, pages 933–938, 2008.

[3] M. Aylett and A. Turk. The Smooth Signal Redundancy Hypothesis: A functional explanation for relationships between redundancy, prosodic prominence, and duration in spontaneous speech. *Language and Speech*, 47:31–56, 2004.

[4] Ted Briscoe. Grammatical Acquisition: Inductive Bias and Coevolution of Language and the Language Acquisition Device. *Language*, 76(2):245–296, 2000.

[5] R. Brown. *A first language*. Harvard University Press, Cambridge, MA, 1973.

[6] Larry Fenson, Philip S. Dale, J. Steven Reznick, Elizabeth Bates, Donna J. Thal, and Stephen J. Pethick. Variability in Early Communicative Development. *Monographs of the Society for Research in Child Development*, 59, 1994.

[7] D. Genzel and E. Charniak. Entropy rate constancy in text. In *In Proceedings of ACL*, 2002.

[8] Talmy Givón. *On Understanding Grammar*. Academic Press, New York, NY, 1979.

[9] R. Hirsh Pasek, K.and Golinkoff. *The origins of grammar: Evidence from early language comprehension*. MIT Press, Cambridge, MA, 1996.

[10] T. F. Jaeger. *Redundancy and syntactic reduction in spontaneous speech*. Unpublished doctoral dissertation, Stanford University, 2006.

[11] T. Florian Jaeger. Redundancy and reduction: Speakers manage syntactic information density. *Cognitive Psychology*, 61:23–62, 2010.

[12] F. Keller. The entropy rate principle as a predictor of processing effort: An evaluation against eye-tracking data. In *Proceedings of the Conference on Empirical Methods in Natural Language Processing*, pages 317–324, 2004.

[13] R. Levy. *Probabilistic Models of Word Order and Syntactic Discontinuity*. PhD thesis, Stanford University, 2005.

[14] R. Levy and T. F. Jaeger. Speakers optimize information density through syntactic reduction. In *Advances in Neural Information Processing Systems*, pages 849–856, 2007.

[15] B. MacWhinney. *The CHILDES project : Tools for analyzing talk*. Lawrence Erlbaum Associates, Mahwah, NJ, 3rd edition, 2000.

[16] B. Miller, P. Hemmer, M. Steyvers, and M.D. Lee. The wisdom of crowds in rank ordering problems. In A. Howesa, D. Peebles, and R. Cooper, editors, *9th International Conference on Cognitive Modeling*, 2009.

[17] Russel S. Tomlin. *Basic word order: functional principles*. Croom Helm, 1986.

